# Adaptive Online Gradient Descent

**Peter L. Bartlett**
Division of Computer Science
Department of Statistics
UC Berkeley
Berkeley, CA 94709
bartlett@cs.berkeley.edu

**Elad Hazan**
IBM Almaden Research Center
650 Harry Road
San Jose, CA 95120
hazan@us.ibm.com

**Alexander Rakhlin** *
Division of Computer Science
UC Berkeley
Berkeley, CA 94709
rakhlin@cs.berkeley.edu

## Abstract

We study the rates of growth of the regret in online convex optimization. First, we show that a simple extension of the algorithm of Hazan *et al* eliminates the need for *a priori* knowledge of the lower bound on the second derivatives of the observed functions. We then provide an algorithm, Adaptive Online Gradient Descent, which interpolates between the results of Zinkevich for linear functions and of Hazan *et al* for strongly convex functions, achieving intermediate rates between $\sqrt{T}$ and $\log T$. Furthermore, we show strong optimality of the algorithm. Finally, we provide an extension of our results to general norms.

## 1 Introduction

The problem of online convex optimization can be formulated as a repeated game between a player and an adversary. At round $t$, the player chooses an action $x_t$ from some convex subset $K$ of $\mathbb{R}^n$, and then the adversary chooses a convex loss function $f_t$. The player aims to ensure that the total loss, $\sum_{t=1}^{T} f_t(x_t)$, is not much larger than the smallest total loss $\sum_{t=1}^{T} f_t(x)$ of any fixed action $x$. The difference between the total loss and its optimal value for a fixed action is known as the *regret*, which we denote

$$\mathcal{R}_T = \sum_{t=1}^{T} f_t(x_t) - \min_{x \in K} \sum_{t=1}^{T} f_t(x).$$

Many problems of online prediction of individual sequences can be viewed as special cases of online convex optimization, including prediction with expert advice, sequential probability assignment, and sequential investment [1]. A central question in all these cases is how the regret grows with the number of rounds of the game.

Zinkevich [2] considered the following gradient descent algorithm, with step size $\eta_t = \Theta(1/\sqrt{t})$. (Here, $\Pi_K(v)$ denotes the Euclidean projection of $v$ on to the convex set $K$.)

---

---
**Algorithm 1** Online Gradient Descent (OGD)
---
1: Initialize $x_1$ arbitrarily.
2: **for** $t = 1$ to $T$ **do**
3:    Predict $x_t$, observe $f_t$.
4:    Update $x_{t+1} = \Pi_K(x_t - \eta_{t+1}\nabla f_t(x_t))$.
5: **end for**
---

Zinkevich showed that the regret of this algorithm grows as $\sqrt{T}$, where $T$ is the number of rounds of the game. This rate cannot be improved in general for arbitrary convex loss functions. However, this is not the case if the loss functions are uniformly convex, for instance, if all $f_t$ have second derivative at least $H > 0$. Recently, Hazan et al [3] showed that in this case it is possible for the regret to grow only *logarithmically* with $T$, using the same algorithm but with the smaller step size $\eta_t = 1/(Ht)$. Increasing convexity makes online convex optimization easier.

The algorithm that achieves logarithmic regret must know in advance a lower bound on the convexity of the loss functions, since this bound is used to determine the step size. It is natural to ask if this is essential: is there an algorithm that can adapt to the convexity of the loss functions and achieve the same regret rates in both cases—$O(\log T)$ for uniformly convex functions and $O(\sqrt{T})$ for arbitrary convex functions? In this paper, we present an adaptive algorithm of this kind.

The key technique is regularization: We consider the online gradient descent (OGD) algorithm, but we add a uniformly convex function, the quadratic $\lambda_t\|x\|^2$, to each loss function $f_t(x)$. This corresponds to shrinking the algorithm's actions $x_t$ towards the origin. It leads to a regret bound of the form

$$\mathcal{R}_T \leq c\sum_{t=1}^{T}\lambda_t + p(\lambda_1, \ldots, \lambda_T).$$

The first term on the right hand side can be viewed as a bias term; it increases with $\lambda_t$ because the presence of the regularization might lead the algorithm away from the optimum. The second term is a penalty for the flatness of the loss functions that becomes smaller as the regularization increases. We show that choosing the regularization coefficient $\lambda_t$ so as to balance these two terms in the bound on the regret up to round $t$ is nearly optimal in a strong sense. Not only does this choice give the $\sqrt{T}$ and $\log T$ regret rates in the linear and uniformly convex cases, it leads to a kind of oracle inequality: The regret is no more than a constant factor times the bound on regret that would have been suffered if an oracle had provided in advance the sequence of regularization coefficients $\lambda_1, \ldots, \lambda_T$ that minimizes the final regret bound.

To state this result precisely, we introduce the following definitions. Let $K$ be a convex subset of $\mathbb{R}^n$ and suppose that $\sup_{x \in K}\|x\| \leq D$. For simplicity, throughout the paper we assume that $K$ is centered around 0, and, hence, $2D$ is the diameter of $K$. Define a shorthand $\nabla_t = \nabla f_t(x_t)$. Let $H_t$ be the largest value such that for any $x^* \in K$,

$$f_t(x^*) \geq f_t(x_t) + \nabla_t^\top(x^* - x_t) + \frac{H_t}{2}\|x^* - x_t\|^2. \tag{1}$$

In particular, if $\nabla^2 f_t - H_t \cdot I \succeq 0$, then the above inequality is satisfied. Furthermore, suppose $\|\nabla_t\| \leq G_t$. Define $\boldsymbol{\lambda}_{1:t} := \sum_{s=1}^{t}\lambda_s$ and $\boldsymbol{H}_{1:t} := \sum_{s=1}^{t}H_s$. Let $\boldsymbol{H}_{1:0} = 0$. Let us now state the Adaptive Online Gradient Descent algorithm as well as the theoretical guarantee for its performance.

---
**Algorithm 2** Adaptive Online Gradient Descent
---
1: Initialize $x_1$ arbitrarily.
2: **for** $t = 1$ to $T$ **do**
3:    Predict $x_t$, observe $f_t$.
4:    Compute $\lambda_t = \frac{1}{2}\left(\sqrt{(\boldsymbol{H}_{1:t} + \boldsymbol{\lambda}_{1:t-1})^2 + 8G_t^2/(3D^2)} - (\boldsymbol{H}_{1:t} + \boldsymbol{\lambda}_{1:t-1})\right)$.
5:    Compute $\eta_{t+1} = (\boldsymbol{H}_{1:t} + \boldsymbol{\lambda}_{1:t})^{-1}$.
6:    Update $x_{t+1} = \Pi_K(x_t - \eta_{t+1}(\nabla f_t(x_t) + \lambda_t x_t))$.
7: **end for**
---

**Theorem 1.1.** *The regret of Algorithm 2 is bounded by*

$$\mathcal{R}_T \leq 3 \inf_{\lambda_1^*,\ldots,\lambda_T^*} \left( D^2 \boldsymbol{\lambda}_{1:T}^* + \sum_{t=1}^{T} \frac{(G_t + \lambda_t^* D)^2}{\boldsymbol{H}_{1:t} + \boldsymbol{\lambda}_{1:t}^*} \right).$$

While Algorithm 2 is stated with the squared Euclidean norm as a regularizer, we show that it is straightforward to generalize our technique to other regularization functions that are uniformly convex with respect to other norms. This leads to adaptive versions of the mirror descent algorithm analyzed recently in [4, 5].

## 2 Preliminary results

The following theorem gives a regret bound for the OGD algorithm with a particular choice of step size. The virtue of the theorem is that the step size can be set without knowledge of the uniform lower bound on $H_t$, which is required in the original algorithm of [3]. The proof is provided in Section 4 (Theorem 4.1), where the result is extended to arbitrary norms.

**Theorem 2.1.** *Suppose we set $\eta_{t+1} = \frac{1}{\boldsymbol{H}_{1:t}}$. Then the regret of OGD is bounded as*

$$\mathcal{R}_T \leq \frac{1}{2} \sum_{t=1}^{T} \frac{G_t^2}{\boldsymbol{H}_{1:t}}.$$

In particular, loosening the bound,

$$2\mathcal{R}_T \leq \frac{\max_t G_t^2}{\min_t \frac{1}{t} \sum_{s=1}^{t} H_s} \log T.$$

Note that nothing prevents $H_t$ from being negative or zero, implying that the same algorithm gives logarithmic regret even when some of the functions are linear or concave, as long as the partial averages $\frac{1}{t} \sum_{s=1}^{t} H_s$ are positive and not too small. The above result already provides an important extension to the log-regret algorithm of [3]: no prior knowledge on the uniform convexity of the functions is needed, and the bound is in terms of the observed sequence $\{H_t\}$. Yet, there is still a problem with the algorithm. If $H_1 > 0$ and $H_t = 0$ for all $t > 1$, then $\sum_{s=1}^{t} H_s = H_1$, resulting in a linear regret bound. However, we know from [2] that a $O(\sqrt{T})$ bound can be obtained. In the next section we provide an algorithm which interpolates between $O(\log T)$ and $O(\sqrt{T})$ bound on the regret depending on the curvature of the observed functions.

## 3 Adaptive Regularization

Suppose the environment plays a sequence of $f_t$'s with curvature $H_t \geq 0$. Instead of performing gradient descent on these functions, we step in the direction of the gradient of $\tilde{f}_t(x) = f_t(x) + \frac{1}{2}\lambda_t \|x\|^2$, where the regularization parameter $\lambda_t \geq 0$ is chosen appropriately at each step as a function of the curvature of the previous functions. We remind the reader that $K$ is assumed to be centered around the origin, for otherwise we would instead use $\|x - x_0\|^2$ to shrink the actions $x_t$ towards the origin $x_0$. Applying Theorem 2.1, we obtain the following result.

**Theorem 3.1.** *If the Online Gradient Descent algorithm is performed on the functions $\tilde{f}_t(x) = f_t(x) + \frac{1}{2}\lambda_t \|x\|^2$ with*

$$\eta_{t+1} = \frac{1}{\boldsymbol{H}_{1:t} + \boldsymbol{\lambda}_{1:t}}$$

*for any sequence of non-negative $\lambda_1, \ldots, \lambda_T$, then*

$$\mathcal{R}_T \leq \frac{1}{2} D^2 \boldsymbol{\lambda}_{1:T} + \frac{1}{2} \sum_{t=1}^{T} \frac{(G_t + \lambda_t D)^2}{\boldsymbol{H}_{1:t} + \boldsymbol{\lambda}_{1:t}}.$$

*Proof.* By Theorem 2.1 applied to functions $\tilde{f}_t$,

$$\sum_{t=1}^{T}\left(f_t(x_t) + \frac{1}{2}\lambda_t\|x_t\|^2\right) \leq \min_x\left(\sum_{t=1}^{T}f_t(x) + \frac{1}{2}\lambda_t\|x\|^2\right) + \frac{1}{2}\sum_{t=1}^{T}\frac{(G_t + \lambda_t D)^2}{\boldsymbol{H}_{1:t} + \boldsymbol{\lambda}_{1:t}}.$$

Indeed, it is easy to verify that condition (1) for $f_t$ implies the corresponding statement with $\tilde{H}_t = H_t + \lambda_t$ for $\tilde{f}_t$. Furthermore, by linearity, the bound on the gradient of $\tilde{f}_t$ is $G_t + \lambda_t\|x_t\| \leq G_t + \lambda_t D$. Define $x^* = \arg\min_x \sum_{t=1}^{T}f_t(x)$. Then, dropping the $\|x_t\|^2$ terms and bounding $\|x^*\|^2 \leq D^2$,

$$\sum_{t=1}^{T}f_t(x_t) \leq \sum_{t=1}^{T}f_t(x^*) + \frac{1}{2}D^2\boldsymbol{\lambda}_{1:T} + \frac{1}{2}\sum_{t=1}^{T}\frac{(G_t + \lambda_t D)^2}{\boldsymbol{H}_{1:t} + \boldsymbol{\lambda}_{1:t}},$$

which proves the the theorem. $\qquad\square$

The following inequality is important in the rest of the analysis, as it allows us to remove the dependence on $\lambda_t$ from the numerator of the second sum at the expense of increased constants. We have

$$\frac{1}{2}\left(D^2\boldsymbol{\lambda}_{1:T} + \sum_{t=1}^{T}\frac{(G_t + \lambda_t D)^2}{\boldsymbol{H}_{1:t} + \boldsymbol{\lambda}_{1:t}}\right) \leq \frac{1}{2}D^2\boldsymbol{\lambda}_{1:T} + \frac{1}{2}\sum_{t=1}^{T}\left(\frac{2G_t^2}{\boldsymbol{H}_{1:t} + \boldsymbol{\lambda}_{1:t}} + \frac{2\lambda_t^2 D^2}{\boldsymbol{H}_{1:t} + \boldsymbol{\lambda}_{1:t-1} + \lambda_t}\right)$$

$$\leq \frac{3}{2}D^2\boldsymbol{\lambda}_{1:T} + \sum_{t=1}^{T}\frac{G_t^2}{\boldsymbol{H}_{1:t} + \boldsymbol{\lambda}_{1:t}}, \qquad (2)$$

where the first inequality holds because $(a + b)^2 \leq 2a^2 + 2b^2$ for any $a, b \in \mathbb{R}$.

It turns out that for appropriate choices of $\{\lambda_t\}$, the above theorem recovers the $O(\sqrt{T})$ bound on the regret for linear functions [2] and the $O(\log T)$ bound for strongly convex functions [3]. Moreover, under specific assumptions on the sequence $\{H_t\}$, we can define a sequence $\{\lambda_t\}$ which produces intermediate rates between $\log T$ and $\sqrt{T}$. These results are exhibited in corollaries at the end of this section.

Of course, it would be nice to be able to choose $\{\lambda_t\}$ adaptively without any restrictive assumptions on $\{H_t\}$. Somewhat surprisingly, such a choice can be made near-optimally by simple local balancing. Observe that the upper bound of Eq. (2) consists of two sums: $D^2\sum_{t=1}^{T}\lambda_t$ and $\sum_{t=1}^{T}\frac{G_t^2}{\boldsymbol{H}_{1:t} + \boldsymbol{\lambda}_{1:t}}$. The first sum increases in any particular $\lambda_t$ and the other decreases. While the influence of the regularization parameters $\lambda_t$ on the first sum is trivial, the influence on the second sum is more involved as all terms for $t \geq t'$ depend on $\lambda_{t'}$. Nevertheless, it turns out that a simple choice of $\lambda_t$ is optimal to within a multiplicative factor of 2. This is exhibited by the next lemma.

**Lemma 3.1.** *Define*

$$H_T(\{\lambda_t\}) = H_T(\lambda_1 \ldots \lambda_T) = \boldsymbol{\lambda}_{1:T} + \sum_{t=1}^{T}\frac{C_t}{\boldsymbol{H}_{1:t} + \boldsymbol{\lambda}_{1:t}},$$

*where $C_t \geq 0$ does not depend on $\lambda_t$'s. If $\lambda_t$ satisfies $\lambda_t = \frac{C_t}{\boldsymbol{H}_{1:t} + \boldsymbol{\lambda}_{1:t}}$ for $t = 1, \ldots, T$, then*

$$H_T(\{\lambda_t\}) \leq 2\inf_{\{\lambda_t^*\}\geq 0}H_T(\{\lambda_t^*\}).$$

*Proof.* We prove this by induction. Let $\{\lambda_t^*\}$ be the optimal sequence of non-negative regularization coefficients. The base of the induction is proved by considering two possibilities: either $\lambda_1 < \lambda_1^*$ or not. In the first case, $\lambda_1 + C_1/(H_1 + \lambda_1) = 2\lambda_1 \leq 2\lambda_1^* \leq 2(\lambda_1^* + C_1/(H_1 + \lambda_1^*))$. The other case is proved similarly.

Now, suppose

$$H_{T-1}(\{\lambda_t\}) \leq 2H_{T-1}(\{\lambda_t^*\}).$$

Consider two possibilities. If $\boldsymbol{\lambda}_{1:T} < \boldsymbol{\lambda}_{1:T}^*$, then

$$H_T(\{\lambda_t\}) = \boldsymbol{\lambda}_{1:T} + \sum_{t=1}^{T}\frac{C_t}{\boldsymbol{H}_{1:t} + \boldsymbol{\lambda}_{1:t}} = 2\boldsymbol{\lambda}_{1:T} \leq 2\boldsymbol{\lambda}_{1:T}^* \leq 2H_T(\{\lambda_t^*\}).$$

If, on the other hand, $\boldsymbol{\lambda}_{1:T} \geq \boldsymbol{\lambda}_{1:T}^*$, then

$$\lambda_T + \frac{C_t}{\boldsymbol{H}_{1:T} + \boldsymbol{\lambda}_{1:T}} = 2\frac{C_t}{\boldsymbol{H}_{1:T} + \boldsymbol{\lambda}_{1:T}} \leq 2\frac{C_t}{\boldsymbol{H}_{1:T} + \boldsymbol{\lambda}_{1:T}^*} \leq 2\left(\lambda_T^* + \frac{C_t}{\boldsymbol{H}_{1:T} + \boldsymbol{\lambda}_{1:T}^*}\right).$$

Using the inductive assumption, we obtain

$$H_T(\{\lambda_t\}) \leq 2H_T(\{\lambda_t^*\}).$$

$\square$

The lemma above is the key to the proof of the near-optimal bounds for Algorithm 2 [1].

*Proof. (of Theorem 1.1)*

By Eq. 2 and Lemma 3.1,

$$\mathcal{R}_T \leq \frac{3}{2}D^2\boldsymbol{\lambda}_{1:T} + \sum_{t=1}^{T}\frac{G_t^2}{\boldsymbol{H}_{1:t} + \boldsymbol{\lambda}_{1:t}} \leq \inf_{\lambda_1^*,\ldots,\lambda_T^*}\left(3D^2\boldsymbol{\lambda}_{1:T}^* + 2\sum_{t=1}^{T}\frac{G_t^2}{\boldsymbol{H}_{1:t} + \boldsymbol{\lambda}_{1:t}^*}\right)$$

$$\leq 6\inf_{\lambda_1^*,\ldots,\lambda_T^*}\left(\frac{1}{2}D^2\boldsymbol{\lambda}_{1:T}^* + \frac{1}{2}\sum_{t=1}^{T}\frac{(G_t + \lambda_t^*D)^2}{\boldsymbol{H}_{1:t} + \boldsymbol{\lambda}_{1:t}^*}\right),$$

provided the $\lambda_t$ are chosen as solutions to

$$\frac{3}{2}D^2\lambda_t = \frac{G_t^2}{\boldsymbol{H}_{1:t} + \boldsymbol{\lambda}_{1:t-1} + \lambda_t}. \tag{3}$$

It is easy to verify that

$$\lambda_t = \frac{1}{2}\left(\sqrt{(\boldsymbol{H}_{1:t} + \boldsymbol{\lambda}_{1:t-1})^2 + 8G_t^2/(3D^2)} - (\boldsymbol{H}_{1:t} + \boldsymbol{\lambda}_{1:t-1})\right)$$

is the non-negative root of the above quadratic equation. We note that division by zero in Algorithm 2 occurs only if $\lambda_1 = H_1 = G_1 = 0$. Without loss of generality, $G_1 \neq 0$, for otherwise $x_1$ is minimizing $f_1(x)$ and regret is negative on that round. $\square$

Hence, the algorithm has a bound on the performance which is 6 times the bound obtained by the best offline adaptive choice of regularization coefficients. While the constant 6 might not be optimal, it can be shown that a constant strictly larger than one is unavoidable (see previous footnote).

We also remark that if the diameter $D$ is unknown, the regularization coefficients $\lambda_t$ can still be chosen by balancing as in Eq. (3), except without the $D^2$ term. This choice of $\lambda_t$, however, increases the bound on the regret suffered by Algorithm 2 by a factor of $O(D^2)$.

Let us now consider some special cases and show that Theorem 1.1 not only recovers the rate of increase of regret of [3] and [2], but also provides intermediate rates. For each of these special cases, we provide a sequence of $\{\lambda_t\}$ which achieves the desired rates. Since Theorem 1.1 guarantees that Algorithm 2 is competitive with the best choice of the parameters, we conclude that Algorithm 2 achieves the same rates.

**Corollary 3.1.** *Suppose $G_t \leq G$ for all $1 \leq t \leq T$. Then for any sequence of convex functions $\{f_t\}$, the bound on regret of Algorithm 2 is $O(\sqrt{T})$.*

*Proof.* Let $\lambda_1 = \sqrt{T}$ and $\lambda_t = 0$ for $1 < t \leq T$. By Eq. 2,

$$\frac{1}{2}\left(D^2\boldsymbol{\lambda}_{1:T} + \sum_{t=1}^{T}\frac{(G_t + \lambda_tD)^2}{\boldsymbol{H}_{1:t} + \boldsymbol{\lambda}_{1:t}}\right) \leq \frac{3}{2}D^2\boldsymbol{\lambda}_{1:T} + \sum_{t=1}^{T}\frac{G_t^2}{\boldsymbol{H}_{1:t} + \boldsymbol{\lambda}_{1:t}}$$

$$\leq \frac{3}{2}D^2\sqrt{T} + \sum_{t=1}^{T}\frac{G^2}{\sqrt{T}} = \left(\frac{3}{2}D^2 + G^2\right)\sqrt{T}.$$

$\square$

Hence, the regret of Algorithm 2 can never increase faster than $\sqrt{T}$. We now consider the assumptions of [3].

**Corollary 3.2.** *Suppose $H_t \geq H > 0$ and $G_t^2 < G$ for all $1 \leq t \leq T$. Then the bound on regret of Algorithm 2 is $O(\log T)$.*

*Proof.* Set $\lambda_t = 0$ for all $t$. It holds that $\mathcal{R}_T \leq \frac{1}{2} \sum_{t=1}^{T} \frac{G_t^2}{\boldsymbol{H}_{1:t}} \leq \frac{1}{2} \sum_{t=1}^{T} \frac{G}{tH} \leq \frac{G}{2H} (\log T + 1)$. $\quad\square$

The above proof also recovers the result of Theorem 2.1. The following Corollary shows a spectrum of rates under assumptions on the curvature of functions.

**Corollary 3.3.** *Suppose $H_t = t^{-\alpha}$ and $G_t \leq G$ for all $1 \leq t \leq T$.*

1. *If $\alpha = 0$, then $\mathcal{R}_T = O(\log T)$.*

2. *If $\alpha > 1/2$, then $\mathcal{R}_T = O(\sqrt{T})$.*

3. *If $0 < \alpha \leq 1/2$, then $\mathcal{R}_T = O(T^\alpha)$.*

*Proof.* The first two cases follow immediately from Corollaries 3.1 and 3.2. For the third case, let $\lambda_1 = T^\alpha$ and $\lambda_t = 0$ for $1 < t \leq T$. Note that $\sum_{s=1}^{t} H_s \geq \int_{x=0}^{t-1} (x+1)^{-\alpha} dx = (1-\alpha)^{-1} t^{1-\alpha} - (1-\alpha)^{-1}$. Hence,

$$\frac{1}{2} \left( D^2 \boldsymbol{\lambda}_{1:T} + \sum_{t=1}^{T} \frac{(G_t + \lambda_t D)^2}{\boldsymbol{H}_{1:t} + \boldsymbol{\lambda}_{1:t}} \right) \leq \frac{3}{2} D^2 \boldsymbol{\lambda}_{1:T} + \sum_{t=1}^{T} \frac{G_t^2}{\boldsymbol{H}_{1:t} + \boldsymbol{\lambda}_{1:t}}$$

$$\leq 2 D^2 T^\alpha + G^2 (1-\alpha) \sum_{t=1}^{T} \frac{1}{t^{1-\alpha} - 1}$$

$$\leq 2 D^2 T^\alpha + 2 G^2 \frac{1}{\alpha} T^\alpha + O(1) = O(T^\alpha).$$

$\square$

## 4 Generalization to different norms

The original online gradient descent (OGD) algorithm as analyzed by Zinkevich [2] used the Euclidean distance of the current point from the optimum as a potential function. The logarithmic regret bounds of [3] for strongly convex functions were also stated for the Euclidean norm, and such was the presentation above. However, as observed by Shalev-Shwartz and Singer in [5], the proof technique of [3] extends to arbitrary norms. As such, our results above for adaptive regularization carry on to the general setting, as we state below . Our notation follows that of Gentile and Warmuth [6].

**Definition 4.1.** *A function $g$ over a convex set $K$ is called $H$-strongly convex with respect to a convex function $h$ if*

$$\forall x, y \in K \; . \; g(x) \geq g(y) + \nabla g(y)^\top (x - y) + \frac{H}{2} B_h(x, y).$$

*Here $B_h(x, y)$ is the Bregman divergence with respect to the function $h$, defined as*

$$B_h(x, y) = h(x) - h(y) - \nabla h(y)^\top (x - y).$$

This notion of strong convexity generalizes the Euclidean notion: the function $g(x) = \|x\|_2^2$ is strongly convex with respect to $h(x) = \|x\|_2^2$ (in this case $B_h(x, y) = \|x - y\|_2^2$). More generally, the Bregman divergence can be thought of as a squared norm, not necessarily Euclidean, i.e., $B_h(x, y) = \|x - y\|^2$. Henceforth we also refer to the *dual norm* of a given norm, defined by $\|y\|_* = \sup_{\|x\| \leq 1} \{y^\top x\}$. For the case of $\ell_p$ norms, we have $\|y\|_* = \|y\|_q$ where $q$ satisfies $\frac{1}{p} + \frac{1}{q} = 1$, and by Hölder's inequality $y^\top x \leq \|y\|_* \|x\| \leq \frac{1}{2} \|y\|_*^2 + \frac{1}{2} \|x\|^2$ (this holds for norms other than $\ell_p$ as well).

For simplicity, the reader may think of the functions $g, h$ as convex and differentiable[2]. The following algorithm is a generalization of the OGD algorithm to general strongly convex functions (see the derivation in [6]). In this extended abstract we state the update rule implicitly, leaving the issues of efficient computation for the full version (these issues are orthogonal to our discussion, and were addressed in [6] for a variety of functions $h$).

---

**Algorithm 3** General-Norm Online Gradient Descent

---

1: Input: convex function $h$
2: Initialize $x_1$ arbitrarily.
3: **for** $t = 1$ to $T$ **do**
4:     Predict $x_t$, observe $f_t$.
5:     Compute $\eta_{t+1}$ and let $y_{t+1}$ be such that $\nabla h(y_{t+1}) = \nabla h(x_t) - 2\eta_{t+1}\nabla f_t(x_t)$.
6:     Let $x_{t+1} = \arg\min_{x \in K} B_h(x, y_{t+1})$ be the projection of $y_{t+1}$ onto $K$.
7: **end for**

---

The methods of the previous sections can now be used to derive similar, dynamically optimal, bounds on the regret. As a first step, let us generalize the bound of [3], as well as Theorem 2.1, to general norms:

**Theorem 4.1.** *Suppose that, for each $t$, $f_t$ is a $H_t$-strongly convex function with respect to $h$, and let $h$ be such that $B_h(x, y) \geq \|x - y\|^2$ for some norm $\|\cdot\|$. Let $\|\nabla f_t(x_t)\|_* \leq G_t$ for all $t$. Applying the General-Norm Online Gradient Algorithm with $\eta_{t+1} = \frac{1}{\boldsymbol{H}_{1:t}}$, we have*

$$\mathcal{R}_T \leq \frac{1}{2}\sum_{t=1}^{T}\frac{G_t^2}{\boldsymbol{H}_{1:t}}.$$

*Proof.* The proof follows [3], with the Bregman divergence replacing the Euclidean distance as a potential function. By assumption on the functions $f_t$, for any $x^* \in K$,

$$f_t(x_t) - f_t(x^*) \leq \nabla f_t(x_t)^\top (x_t - x^*) - \frac{H_t}{2}B_h(x^*, x_t).$$

By a well-known property of Bregman divergences (see [6]), it holds that for any vectors $x, y, z$,

$$(x - y)^\top (\nabla h(z) - \nabla h(y)) = B_h(x, y) - B_h(x, z) + B_h(y, z).$$

Combining both observations,

$$
\begin{aligned}
2(f_t(x_t) - f_t(x^*)) &\leq 2\nabla f_t(x_t)^\top (x_t - x^*) - H_t B_h(x^*, x_t) \\
&= \frac{1}{\eta_{t+1}}(\nabla h(y_{t+1}) - \nabla h(x_t))^\top (x^* - x_t) - H_t B_h(x^*, x_t) \\
&= \frac{1}{\eta_{t+1}}[B_h(x^*, x_t) - B_h(x^*, y_{t+1}) + B_h(x_t, y_{t+1})] - H_t B_h(x^*, x_t) \\
&\leq \frac{1}{\eta_{t+1}}[B_h(x^*, x_t) - B_h(x^*, x_{t+1}) + B_h(x_t, y_{t+1})] - H_t B_h(x^*, x_t),
\end{aligned}
$$

where the last inequality follows from the Pythagorean Theorem for Bregman divergences [6], as $x_{t+1}$ is the projection w.r.t the Bregman divergence of $y_{t+1}$ and $x^* \in K$ is in the convex set. Summing over all iterations and recalling that $\eta_{t+1} = \frac{1}{\boldsymbol{H}_{1:t}}$,

$$
\begin{aligned}
2\mathcal{R}_T &\leq \sum_{t=2}^{T} B_h(x^*, x_t)\left(\frac{1}{\eta_{t+1}} - \frac{1}{\eta_t} - H_t\right) + B_h(x^*, x_1)\left(\frac{1}{\eta_2} - H_1\right) + \sum_{t=1}^{T}\frac{1}{\eta_{t+1}}B_h(x_t, y_{t+1}) \\
&= \sum_{t=1}^{T}\frac{1}{\eta_{t+1}}B_h(x_t, y_{t+1}). \tag{4}
\end{aligned}
$$

We proceed to bound $B_h(x_t, y_{t+1})$. By definition of Bregman divergence, and the dual norm inequality stated before,

$$B_h(x_t, y_{t+1}) + B_h(y_{t+1}, x_t) = (\nabla h(x_t) - \nabla h(y_{t+1}))^\top (x_t - y_{t+1})$$
$$= 2\eta_{t+1} \nabla f_t(x_t)^\top (x_t - y_{t+1})$$
$$\leq \eta_{t+1}^2 \|\nabla_t\|_*^2 + \|x_t - y_{t+1}\|^2.$$

Thus, by our assumption $B_h(x, y) \geq \|x - y\|^2$, we have

$$B_h(x_t, y_{t+1}) \leq \eta_{t+1}^2 \|\nabla_t\|_*^2 + \|x_t - y_{t+1}\|^2 - B_h(y_{t+1}, x_t) \leq \eta_{t+1}^2 \|\nabla_t\|_*^2.$$

Plugging back into Eq. (4) we get

$$\mathcal{R}_T \leq \frac{1}{2} \sum_{t=1}^{T} \eta_{t+1} G_t^2 = \frac{1}{2} \sum_{t=1}^{T} \frac{G_t^2}{\boldsymbol{H}_{1:t}}.$$

$\square$

The generalization of our technique is now straightforward. Let $A^2 = \sup_{x \in K} g(x)$ and $2B = \sup_{x \in K} \|\nabla g(x)\|_*$. The following algorithm is an analogue of Algorithm 2 and Theorem 4.2 is the analogue of Theorem 1.1 for general norms.

---

**Algorithm 4** Adaptive General-Norm Online Gradient Descent

1: Initialize $x_1$ arbitrarily. Let $g(x)$ be 1-strongly convex with respect to the convex function $h$.
2: **for** $t = 1$ to $T$ **do**
3:     Predict $x_t$, observe $f_t$
4:     Compute $\lambda_t = \frac{1}{2} \left( \sqrt{(\boldsymbol{H}_{1:t} + \boldsymbol{\lambda}_{1:t-1})^2 + 8G_t^2/(A^2 + 2B^2)} - (\boldsymbol{H}_{1:t} + \boldsymbol{\lambda}_{1:t-1}) \right).$
5:     Compute $\eta_{t+1} = (\boldsymbol{H}_{1:t} + \boldsymbol{\lambda}_{1:t})^{-1}.$
6:     Let $y_{t+1}$ be such that $\nabla h(y_{t+1}) = \nabla h(x_t) - 2\eta_{t+1}(\nabla f_t(x_t) + \frac{\lambda_t}{2} \nabla g(x_t))).$
7:     Let $x_{t+1} = \arg\min_{x \in K} B_h(x, y_{t+1})$ be the projection of $y_{t+1}$ onto $K$.
8: **end for**

---

**Theorem 4.2.** *Suppose that each $f_t$ is a $H_t$-strongly convex function with respect to $h$, and let $g$ be a 1-strongly convex with respect $h$. Let $h$ be such that $B_h(x, y) \geq \|x - y\|^2$ for some norm $\|\cdot\|$. Let $\|\nabla f_t(x_t)\|_* \leq G_t$. The regret of Algorithm 4 is bounded by*

$$\mathcal{R}_T \leq \inf_{\lambda_1^*, \ldots, \lambda_T^*} \left( (A^2 + 2B^2)\boldsymbol{\lambda}_{1:T}^* + \sum_{t=1}^{T} \frac{(G_t + \lambda_t^* B)^2}{\boldsymbol{H}_{1:t} + \boldsymbol{\lambda}_{1:t}^*} \right).$$

If the norm in the above theorem is the Euclidean norm and $g(x) = \|x\|^2$, we find that $D = \sup_{x \in K} \|x\| = A = B$ and recover the results of Theorem 1.1.

## Footnotes

[1]Lemma 3.1 effectively describes an algorithm for an online problem with competitive ratio of 2. In the full version of this paper we give a lower bound strictly larger than one on the competitive ratio achievable by any online algorithm for this problem.

[2]Since the set of points of nondifferentiability of convex functions has measure zero, convexity is the only property that we require. Indeed, for nondifferentiable functions, the algorithm would choose a point $\tilde{x}_t$, which is $x_t$ with the addition of a small random perturbation. With probability one, the functions would be smooth at the perturbed point, and the perturbation could be made arbitrarily small so that the regret rate would not be affected.

## References

[1] Nicolò Cesa-Bianchi and Gábor Lugosi. *Prediction, Learning, and Games*. Cambridge University Press, 2006.

[2] Martin Zinkevich. Online convex programming and generalized infinitesimal gradient ascent. In *ICML*, pages 928–936, 2003.

[3] Elad Hazan, Adam Kalai, Satyen Kale, and Amit Agarwal. Logarithmic regret algorithms for online convex optimization. In *COLT*, pages 499–513, 2006.

[4] Shai Shalev-Shwartz and Yoram Singer. Convex repeated games and Fenchel duality. In B. Schölkopf, J. Platt, and T. Hoffman, editors, *Advances in Neural Information Processing Systems 19*. MIT Press, Cambridge, MA, 2007.

[5] Shai Shalev-Shwartz and Yoram Singer. Logarithmic regret algorithms for strongly convex repeated games. In *Technical Report 2007-42*. The Hebrew University, 2007.

[6] C. Gentile and M. K. Warmuth. Proving relative loss bounds for on-line learning algorithms using Bregman divergences. In *COLT. Tutorial*, 2000.

